# A Superaddidive-Impairment Theory of Optic Aphasia

**Michael C. Mozer**
*Dept. of Computer Science*
*University of Colorado*
*Boulder, CO 80309–0430*

**Mark Sitton**
*Dept. of Computer Science*
*University of Colorado*
*Boulder, CO 80309–0430*

**Martha Farah**
*Dept. of Psychology*
*University of Pennsylvania*
*Phila., PA 19104–6196*

## Abstract

Accounts of neurological disorders often posit damage to a specific functional pathway of the brain. Farah (1990) has proposed an alternative class of explanations involving partial damage to multiple pathways. We explore this explanation for *optic aphasia*, a disorder in which severe performance deficits are observed when patients are asked to name visually presented objects, but surprisingly, performance is relatively normal on naming objects from auditory cues and on gesturing the appropriate use of visually presented objects. We model this highly specific deficit through partial damage to two pathways—one that maps visual input to semantics, and the other that maps semantics to naming responses. The effect of this damage is *superadditive*, meaning that tasks which require one pathway or the other show little or no performance deficit, but the damage is manifested when a task requires both pathways (i.e., naming visually presented objects). Our model explains other phenomena associated with optic aphasia, and makes testable experimental predictions.

Neuropsychology is the study of disrupted cognition resulting from damage to functional systems in the brain. Generally, accounts of neuropsychological disorders posit damage to a particular functional system or a disconnection between systems. Farah (1990) suggested an alternative class of explanations for neuropsychological disorders: partial damage to multiple systems, which is manifested through interactions among the loci of damage. We explore this explanation for the neuropsychological disorder of *optic aphasia*.

Optic aphasia, arising from unilateral left posterior lesions, including occipital cortex and the splenium of the corpus callosum (Schnider, Benson, & Scharre, 1994), is marked by a deficit in naming visually presented objects, hereafter referred to as *visual naming* (Farah, 1990). However, patients can demonstrate recognition of visually presented objects nonverbally, for example, by gesturing the appropriate use of an object or sorting visual items into their proper superordinate categories (hereafter, *visual gesturing*). Patients can also name objects by nonvisual cues such as a verbal definition or typical sounds made by the objects (hereafter, *auditory naming*). The highly specific nature of the deficit rules out an explanation in terms of damage to a single pathway in a standard model of visual naming (Figure 1), suggesting that a more complex model is required, involving

**FIGURE 1. A standard box-and-arrow model of visual naming. The boxes denote levels of representation, and the arrows denote pathways mapping from one level of representation to another. Although optic aphasia cannot be explained by damage to the vision-to-semantics pathway or the semantics-to-naming pathway, Farah (1990) proposed an explanation in terms of partial damage to both pathways (the X's).**

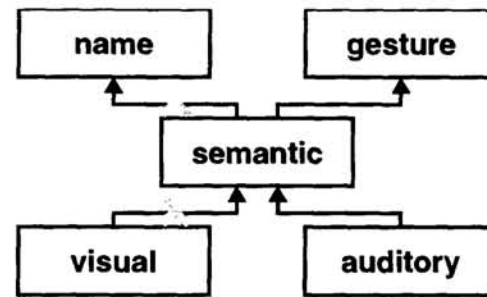

multiple semantic systems or multiple pathways to visual naming. However, a more parsimonious account is suggested by Farah (1990): Optic aphasia might arise from *partial* lesions to two pathways in the standard model—those connecting visual input to semantics, and semantics to naming—and the effect of damage to these pathways is *superadditive*, meaning that tasks which require only one of these pathways (e.g., visual gesturing, or auditory naming) will be relatively unimpaired, whereas tasks requiring both pathways (e.g., visual naming) will show a significant deficit.

# 1 A MODEL OF SUPERADDITIVE IMPAIRMENTS

We present a computational model of the superadditive-impairment theory of optic aphasia by elaborating the architecture of Figure 1. The architecture has four pathways: visual input to semantics (V→S), auditory input to semantics (A→S), semantics to naming (S→N), and semantics to gesturing (S→G). Each pathway acts as an associative memory. The critical property of a pathway that is required to explain optic aphasia is a *speed-accuracy trade off*: The initial output of a pathway appears rapidly, but it may be inaccurate. This "quick and dirty" guess is refined over time, and the pathway output asymptotically converges on the best interpretation of the input.

We implement a pathway using the architecture suggested by Mathis and Mozer (1996). In this architecture, inputs are mapped to their best interpretations by means of a two-stage process (Figure 2). First, a quick, one-shot *mapping* is performed by a multi-layer feedforward connectionist network to transform the input directly to its corresponding output. This is followed by a slower iterative *clean-up* process carried out by a recurrent attractor network. This architecture shows a speed-accuracy trade off by virtue of the assumption that the feedforward mapping network does not have the capacity to produce exactly the right output to every input, especially when the inputs are corrupted by noise or are otherwise incomplete. Consequently, the clean up stage is required to produce a sensible interpretation of the noisy output of the mapping network.

Fully distributed attractor networks have been used for similar purposes (e.g., Plaut & Shallice, 1993). For simplicity, we adopt a localist-attractor network with a layer of *state* units and a layer of radial basis function (RBF) units, one RBF unit per attractor. Each RBF or *attractor* unit measures the distance of the current state to the attractor that it represents. The activity of attractor unit $i$, $a_i$, is:

**FIGURE 2. Connectionist implementation of a processing pathway. The pathway consists of feedforward *mapping* network followed by a recurrent *clean-up* or attractor network. Circles denote connectionist processing units and arrows denote connections between units or between layers of units.**

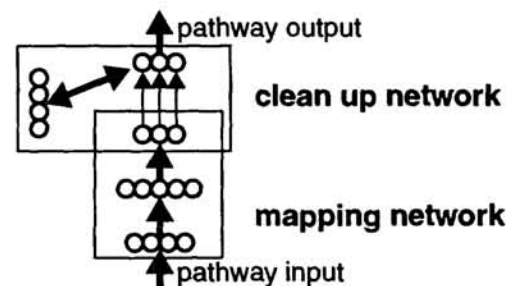

$$\hat{a}_i(t) = \exp(-\|s(t) - \mu_i\|^2 / \beta_i) \tag{1}$$

$$a_i(t) = \frac{\hat{a}_i(t)}{\sum_j \hat{a}_j(t)} \tag{2}$$

where $s(t)$ is the state unit activity vector at time $t$, $\mu_i$ is the vector denoting the location of attractor $i$, and $\beta_i$ is the *strength* of the attractor. The strength determines the region of the state space over which an attractor will exert its pull, and also the rate at which the state will converge to the attractor. The state units receive input from the mapping network and from the attractor units and are updated as follows:

$$s_i(t) = d_i(t)e_i(t) + (1 - d_i(t))\sum_j a_j(t-1)\mu_{ji} \tag{3}$$

where $s_i(t)$ is the activity of state unit $i$ at time $t$, $e_i$ is the $i$th output of the mapping net, $\mu_{ji}$ is the $i$th element of attractor $j$, and $d_i$ is given by

$$d_i(t) = h\left[1 - \frac{\bar{e}_i(t-1)}{e_i(t)}\right] \tag{4}$$

where $h[.]$ is a linear threshold function that bounds activity between $-1$ and $+1$, $\bar{e}_i$ is a weighted time average of the $i$th output of the mapping net,

$$\bar{e}_i(t) = \alpha e_i(t) + (1 - \alpha)\bar{e}_i(t-1) \tag{5}$$

In all simulations, $\alpha = .02$.

The activity of the state units are governed by two forces—the external input from the feedforward net (first term in Equation 3) and the attractor unit activities (second term). The parameter $d_i$ acts as a kind of attentional mechanism that modulates the relative influence of these two forces. The basic idea is that when the input coming from the mapping net is changing, the system should be responsive to the input and should not yet be concerned with interpreting the input. In this case, the input is copied straight through to the state units and hence $d_i$ should have a value close to 1. When the input begins to stabilize, however, the focus shifts to interpreting the input and following the dynamics of the attractor network. This shift corresponds to $d_i$ being lowered to zero. The weighted time average in the update rule for $d_i$ is what allows for the smooth transition of the function to its new value. For certain constructions of the function $d$, Zemel and Mozer (in preparation) have proven convergence of the algorithm to an attractor.

Apart from speed-accuracy trade off, these dynamics have another important consequence for the present model, particularly with respect to cascading pathways. If pathway A feeds into pathway B, such as V→S feeding into S→N, then the state unit activities of A act as the input to B. Because these activities change over time as the state approaches a well-formed state, the dynamics of pathway B can be quite complex as it is forced to deal with an unstable input. This property is important in explaining several phenomena associated with optic aphasia.

## 1.1 PATTERN GENERATION

Patterns were constructed for each of the five representational spaces: *visual* and *auditory* input, *semantic*, *name* and *gesture* responses. Each representational space was arbitrarily made to be 200 dimensional. We generated 200 binary-valued $(-1,+1)$ patterns in each space, which were meant to correspond to known entities of that representational domain.

For the visual, auditory, and semantic spaces, patterns were partitioned into 50 similarity *clusters* with 4 *siblings* per cluster. Patterns were chosen randomly subject to two constraints: patterns in different clusters had to be at least 80° apart, and siblings had to be between 25° and 50° apart. Because similarity of patterns in the name and gesture spaces was irrelevant to our modeling, we did not impose a similarity structure on these spaces.

Instead, we generated patterns in these spaces at random subject to the constraint that every pattern had to be at least 60° from every other.

After generating patterns in each of the representational spaces, we established arbitrary correspondences among the patterns such that visual pattern *n*, auditory pattern *n*, semantic pattern *n*, name pattern *n*, and gesture pattern *n* all represented the same concept. That is, the appropriate response in a visual-naming task to visual pattern *n* would be semantic pattern *n* and name pattern *n*.

## 1.2 TRAINING PROCEDURE

The feedforward networks in the four pathways (V→S, A→S, S→N, and S→G) were independently trained on all 200 associations using back propagation. Each of these networks contained a single hidden layer of 150 units, and all units in the network used the symmetric activation function to give activities in the range [−1,+1]. The amount of training was chosen such that performance on the training examples was not perfect; usually several elements in the output would be erroneous—i.e., have the wrong sign—and others would not be exactly correct—i.e., −1 or +1. This was done to embody the architectural assumption that the feedforward net does not have the capacity to map every input to exactly the right output, and hence, the clean-up process is required.

Training was not required for the clean-up network. Due to the localist representation of attractors in the clean-up network, it was trivial to hand wire each clean-up net with the 200 attractors for its domain, along with one *rest-state* attractor. All attractor strengths were initialized to the same value, $\beta=15$, except the rest-state attractor, for which $\beta=5$. The rest-state attractor required a lower strength so that even a weak external input would be sufficient to kick the attractor network out of the rest state.

## 1.3 SIMULATION METHODOLOGY

After each pathway had been trained, the model was damaged by "lesioning" or removing a fraction $\gamma$ of the connections in the V→S and S→N mapping networks. The lesioned connections were chosen at random and an equal fraction was removed from the two pathways. The clean-up nets were not damaged. The architecture was damaged a total of 30 different times, creating 30 simulated patients who were tested on each of the four tasks and on all 200 input patterns for a task. The results we report come from averaging across simulated patients and input patterns. Responses were determined after the system had been given sufficient time to relax into a name or gesture attractor, which was taken to be the response. Each response was classified as one of the following mutually exclusive response types: *correct*, *perseveration* (response is the same as that produced on any of the three immediately preceding trials), *visual error* (the visual pattern corresponding to the incorrect response is a sibling of the visual pattern corresponding to the correct response), *semantic error*, *visual+semantic error*, or *other error*.

## 1.4 PRIMING MECHANISM

*Priming*—the increased availability of recently experienced stimuli—has been found across a wide variety of tasks in normal subjects. We included priming in our model as a strengthening (increasing the $\beta_i$ parameter) of recently visited attractors (see Mathis & Mozer 1996, for details, and Becker, Behrmann, & Moscovitch, 1993, for a related approach). In the damaged model, this mechanism often gave rise to perseverations.

## 2 RESULTS

We have examined the model's behavior as we varied the amount of damage, quantified by the parameter $\gamma$. However, we report on the performance of simulated patients with $\gamma = .30$. This intermediate amount of damage yielded no floor or ceiling effects, and also produced error rates for the visual-naming task in the range of 30-40%, roughly the median performance of patients in the literature.

**TABLE 1. Error rate of the damaged model on various tasks.**

| task | error rate |
|---|---|
| auditory gesturing | 0.0% |
| auditory naming | 0.5% |
| visual gesturing | 8.7% |
| visual naming | 36.8% |

Table 1 presents the error rates of the model on four tasks. The pattern of errors shows a qualitative fit to human patient data. The model produced no errors on the auditory gesturing task because the two component pathways (A→S and S→G) were undamaged. Relatively few errors were made on the auditory-naming and visual-gesturing tasks, each of which involved one damaged pathway, because the clean-up nets were able to compensate for the damage. However, the error rate for the visual-naming task was quite large, due to damage on both of its component pathways (V→S and S→N). The error rate for visual naming cannot be accounted for by summing the effects of the damage to the two component pathways because the sum of the error rates for auditory naming and visual gesturing, each of which involves one of the two partially damaged pathways, is nearly four times smaller. Rather, the effects of damage on these pathways interact, and their interaction leads to superadditive impairments.

When a visual pattern is presented to the model, it is mapped by the damaged V→S pathway into a corrupted semantic representation which is then cleaned up. While the corruption is sufficiently minor that clean up will eventually succeed, the clean up process is slowed considerably by the corruption. During the period of time in which the semantic clean-up network is searching for the correct attractor, the corrupted semantic representation is nonetheless fed into the damaged S→N pathway. The combined effect of the (initially) noisy semantic representation serving as input to a damaged pathway leads to corruption of the naming representation past the point where it can be cleaned-up properly.

Interactions in the architecture are inevitable, and are not merely a consequence of some arbitrary assumption that is built into our model. To argue this point, we consider two modifications to the architecture that might eliminate the interaction in the damaged model. First, if we allowed the V→S pathway to relax into a well-formed state before feeding its output into the S→N pathway, there would be little interaction—the effects of the damage would be additive. However, cortical pathways do not operate sequentially, one stage finishing its computation and then turning on the next stage. Moreover, in the undamaged brain, such a processing strategy is unadaptive, as cascading partial results from one pathway to the next can speed processing without the introduction of errors (McClelland, 1979). Second, the interaction might be eliminated by making the S→N pathway continually responsive to changes in the output of the V→S pathway. Then, the rate of convergence of the V→S pathway would be irrelevant to determining the eventual output of the S→N pathway. However, because the output of the S→N pathway depends not only on its input but its internal state (the state of the clean-up net), one cannot design a pathway that is continually responsive to changes in the input and is also able to clean up noisy responses. Thus, the two modifications one might consider to eliminate the interactions in the damaged model seriously weaken the computational power of the undamaged model. We therefore conclude that the framework of our model makes it difficult to avoid an interaction of damage in two pathways.

A subtle yet significant aspect of the model's performance is that the error rate on the visual-gesturing task was reliably higher than the error rate on the auditory-naming task, despite the fact that each task made use of one damaged pathway, and the pathways were damaged to the same degree. The difference in performance is due to the fact that the damaged pathway for the visual-gesturing task is the first in a cascade of two, while the damaged pathway for the auditory-naming task is the second. The initially noisy response from a damaged pathway early in the system propagates to subsequent pathways, and

although the damaged pathway will eventually produce the correct response, this is not sufficient to ensure that subsequent pathways will do so as well.

## 2.1  DISTRIBUTION OF ERRORS FOR VISUAL OBJECT NAMING

Figure 2 presents the model's error distribution for the visual-naming task. Consistent with the patient data (Farah, 1990), the model produces many more semantic and perseveration errors than by chance. The chance error proportions were computed by assuming that if the correct response was not made, then all other responses had an equal probability of being chosen.

To understand the predominance of semantic errors, consider the effect of damage to the V→S pathway. For relatively small amounts of damage, the mapping produced will be close to the correct mapping. "Close" here means that the Euclidean distance in the semantic output space between the correct and perturbed mapping is small. Most of the time, minor perturbation of the mapping will be compensated for by the clean-up net. Occasionally, the perturbation will land the model in a different attractor basin, and a different response will be made. However, when the wrong attractor is selected, it will be one "close" to the correct attractor, i.e., it will likely be a sibling in the same pattern cluster as the correct attractor. In the case of the V→S pathway, the siblings of the correct attractor are by definition semantically related. A semantic error will be produced by the model when a sibling semantic attractor is chosen, and then this pattern is correctly mapped to a naming response in the S→N pathway.

In addition to semantic errors, the other frequent error type in visual naming is perseverations. The priming mechanism is responsible for the significant number of perseverations, although in the unlesioned model, it facilitates processing of repeated stimuli without producing perseverations.

Just as important as the presence of perseverative and semantic errors is the absence of visual errors, a feature of optic aphasia that contrasts sharply with visual agnosia (Farah, 1990). The same mechanisms explain why the rate of visual errors is close to its chance value and why visual+semantic errors are above chance. Visual-naming errors occur because there is an error either in the V→S or S→N mappings, or both. Since the erroneous outputs of these pathways show a strong tendency to be similar to the correct output, and because semantic and name similarity does not imply visual similarity (the patterns were paired randomly), visual errors should only occur by chance. When a visual error does occur, though, there is a high probability that the error is also semantic because of the strong bias that already exists toward producing semantic errors. This is the reason why more visual+semantic errors occur than by chance and why the proportion of these

**FIGURE 3. Distribution of error types made by model on the V→N task (black bars) relative to chance (grey bars).**

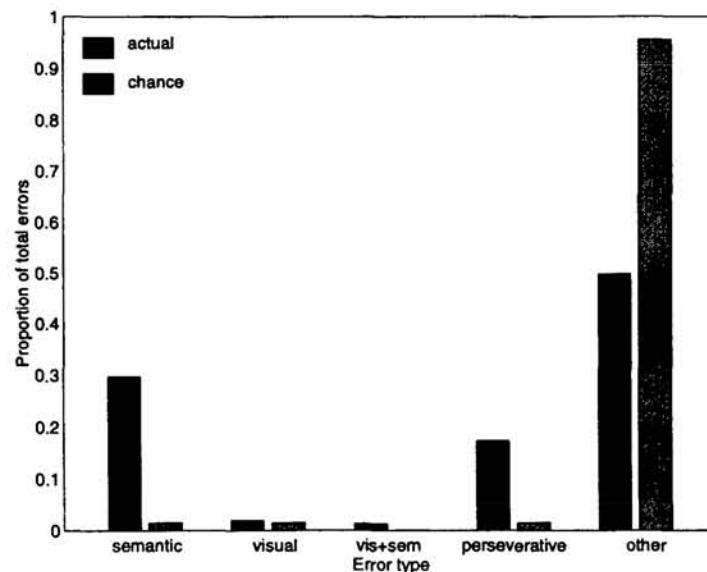

errors is only slightly less than the proportion of visual errors.

Plaut and Shallice (1993) have proposed a connectionist model to account for the distribution of errors made by optic aphasics. Although their model was not designed to account for any of the other phenomena associated with the disorder, it has much in common with the model we are proposing. Unlike our model, however, theirs requires the assumption that visually similar objects also share semantic similarity. This assumption might be questioned, especially because our model does not require this assumption to produce the correct distribution of error responses.

## 3  DISCUSSION

In demonstrating superadditive effects of damage, we have offered an account of optic aphasia that explains the primary phenomenon: severe impairments in visual naming in conjunction with relatively spared performance on naming from verbal description or gesturing the appropriate use of a visually presented object. The model also explains the distribution of errors on visual naming. Although we did not have the space in this brief report to elaborate, the model accounts for several other distinct characteristics of optic aphasia, including the tendency of patients to "home in" on the correct name for a visually presented object when given sufficient time, and a positive correlation between the error rates on naming and gesturing responses to a visual object (Sitton, Mozer, & Farah, 1998). Further, the model makes several strong predictions which have yet to be tested experimentally. One such prediction, which was apparent in the results presented earlier, is that a higher error rate should be observed on visual gesturing than on auditory naming when the tasks are equated for difficulty, as our simulation does.

More generally, we have strengthened the plausibility of Farah's (1990) hypothesis that partial damage to two processing pathways may result in close-to-normal performance on tasks involving one pathway or the other while yielding a severe performance deficit on tasks involving both damaged pathways. The superadditive-impairment theory thus may provide a more parsimonious account of various disorders that were previously believed to require more complex architectures or explanations.

## 4  ACKNOWLEDGMENTS

This research was supported by grant 97-18 from the McDonnell-Pew Program in Cognitive Neuroscience.

## 5  REFERENCES

Becker, S., Behrmann, M., & Moscovitch, K. (1993). Word priming in attractor networks. *Proceedings of the Fifteenth Annual Conference of the Cognitive Science Society* (pp. 231–236). Hillsdale, NJ: Erlbaum.

Farah, M. J. (1990). *Visual agnosia*. Cambridge, MA: MIT Press/Bradford Books.

Mathis, D. W., & Mozer, M. C. (1996). Conscious and unconscious perception: A computational theory. In G. Cottrell (Ed.), *Proceedings of the Eighteenth Annual Conference of the Cognitive Science Society* (pp. 324–328). Hillsdale, NJ: Erlbaum.

McClelland, J. L. (1979). On the time relations of mental processes: An examination of systems of processes in cascade. *Psychological Review, 86*, 287–330.

Plaut, D., & Shallice, T. (1993). Perseverative and semantic influences on visual object naming errors in optic aphasia: A connectionist approach. *Journal of Cognitive Neuroscience, 5*, 89-112.

Schnider, A., Benson, D. F., and Scharre, D. W. (1994). Visual agnosia and optic aphasia: Are they anatomically distinct? *Cortex, 30*, 445-457.

Sitton, M., Mozer, M. C., & Farah, M. (1998). *Diffuse lesions in a modular connectionist architecture: An account of optic aphasia*. Manuscript submitted for publication.